# Locally Uniform Comparison Image Descriptor

**Andrew Ziegler**[*]   **Eric Christiansen**   **David Kriegman**   **Serge Belongie**
Department of Computer Science and Engineering, University of California, San Diego
`amz@gatech.edu`, {`echristiansen, kriegman, sjb`}`@cs.ucsd.edu`

## Abstract

Keypoint matching between pairs of images using popular descriptors like SIFT or a faster variant called SURF is at the heart of many computer vision algorithms including recognition, mosaicing, and structure from motion. However, SIFT and SURF do not perform well for real-time or mobile applications. As an alternative very fast binary descriptors like BRIEF and related methods use pairwise comparisons of pixel intensities in an image patch. We present an analysis of BRIEF and related approaches revealing that they are hashing schemes on the ordinal correlation metric Kendall's tau. Here, we introduce Locally Uniform Comparison Image Descriptor (LUCID), a simple description method based on linear time permutation distances between the ordering of RGB values of two image patches. LUCID is computable in linear time with respect to the number of pixels and does not require floating point computation.

## 1   Introduction

Local image descriptors have long been explored in the context of machine learning and computer vision. There are countless applications that rely on local feature descriptors, such as visual registration, reconstruction and object recognition. One of the most widely used local feature descriptors is SIFT which uses automatic scale selection, orientation normalization, and histograms of oriented gradients to achieve partial affine invariance [15]. SIFT is known for its versatility and reliable recognition performance, but these characteristics come at a high computational cost.

Recently, mobile devices and affordable reliable imaging sensors have become ubiquitous. The wide adoption of these devices has made new real-time mobile applications of computer vision and machine learning feasible. Examples of such applications include visual search, augmented reality, perceptual interfaces, and wearable computing. Despite this, these devices have less computational power than typical computers and perform poorly for floating point heavy applications. These factors have provided an impetus for new efficient discrete approaches to feature description and matching. In this work we explore current trends in feature description and provide a new view of BRIEF and its related methods. We also present a novel feature description method that is surprisingly simple and effective.

### 1.1   Background

Bay et al. proposed SURF as an approximation to SIFT, a notable shift toward real-time feature description [1]. SURF obtains a large speed up over SIFT while retaining most of its desirable properties and comparable recognition rates. However, SURF is not generally suited to real-time applications without acceleration via a powerful GPU [21].

In [3] Bosch et al. proposed Ferns as a classification based approach to key point recognition. Ferns uses sparse binary intensity comparisons between pixels in an image patch for descriptive power.

---

[*]This work was completed while the author was at UCSD.

This simple scheme provides real-time performance in exchange for expensive off-line learning. In response to the success of Ferns, Calonder et al. presented a novel binary feature descriptor they named BRIEF [4]. Rather than training off-line, BRIEF makes use of random pixel intensity comparisons to create a binary descriptor quickly. These descriptors can be matched an order of magnitude faster than SIFT with the Hamming distance, even on mobile processors. As a result, BRIEF has come into widespread use and has inspired several variants based on the approach [12, 14, 19]. However, little explanation as to why or how these types of descriptors work is given. There is a fuzzy notion that pairwise intensity comparisons are an approximation to signed intensity gradients. This is not the whole story, and in fact these methods are sampling in an ad hoc manner from a rich source of discriminative information.

## 1.2 Related work

In this work we diverge from the current paradigm for fast feature description and explore a deterministic approach based on permutations. The study of distances between permutations began near the inception of group theory and has continued unabated since [5, 7, 8, 9, 11, 10, 16].

A notable early use of permutation based methods in the realm of visual feature description was presented by Bhat and Nayar in [2]. They investigated the use of rank permutations of pixel intensities for the purpose of dense stereo, the motivation being to find a robust alternative to the $\ell_2$ norm. Permutations on pixel intensities offer a transformed representation of the data which is naturally less sensitive to noise and invariant to monotonic photometric transformations. Bhat and Nayar present a similarity measure between two rank permutations that is based on the Kolmogorov Smirnov test. Their measure was designed to be robust to impulse noise, sometimes called salt and pepper noise, which can greatly corrupt a rank permutation. In [20] Scherer et al. reported that though Bhat and Nayar's method was useful, it suffered from poor discrimination.

In [18] Mittal and Ramesh proposed an improved version of the method presented by Bhat and Nayar. Their improvement was in a similar vein to [20], based on a modification to Kendall's tau [11]. The key observation made was that both Kendall's tau metric and Bhat and Nayar's metric are highly sensitive to Gaussian noise. To become robust to Gaussian noise Mittal and Ramesh account for actual intensity differences while only considering uncorrelated order changes. We choose to explore the Hamming and Cayley distances, in part because they are naturally robust to Gaussian noise, impulse noise is not a major issue for modern imaging devices, and they are computable in linear time as opposed to quadratic time.

Recently there has been more research on the application of ordinal correlation methods to sparse visual feature description. In [22] and [13] ordinal methods were applied to SIFT descriptors. In contrast to [2] and [20] the elements of the SIFT descriptor are sorted, rather than sorting pixel intensities themselves. Though these methods do improve the recognition performance of SIFT they add computational cost, rather than reducing it.

## 1.3 Our contributions

In this paper, we introduce LUCID, a novel approach to real-time feature description based on order permutations. We contrast LUCID with BRIEF, and provide a theoretical basis for understanding these two methods. We prove that BRIEF is effectively a locality sensitive hashing (LSH) scheme on Kendall's tau. It follows from this that other descriptors based on binary intensity comparisons are dimensionality reduction schemes on Kendall's tau. We then explore alternative distances based on the observation that image patch matching can be viewed as a near duplicate recognition problem.

In the next section we describe LUCID, provide a background on permutation distances and discuss optimizations for an efficient implementation. Section 3 provides an analysis of BRIEF and compares it to LUCID. Section 4 reports on experiments that evaluate LUCID's accuracy and run time relative to SURF and BRIEF.

## 2 LUCID

Here we present a new method of feature description that is surprisingly simple and effective. We call our method Locally Uniform Comparison Image Descriptor or LUCID. Our descriptors implicitly encapsulate all possible intensity comparisons in a local area of an image. They are extremely efficient to compute and are related through the generalized Hamming distance for efficient matching [10].

### 2.1 Constructing a descriptor

Let $\mathbf{p_1}$ and $\mathbf{p_2}$ be $n \times n$ image patches with $c$ color channels. We can compute descriptors for both patches and the Hamming distance between them in three lines of Matlab as shown in Figure 1. Here **desc1** and **desc2** are the order permutation representations for **p1** and **p2** respectively. Let $m = cn^2$, then clearly this depiction has an $O(m \log m)$ running time. However, our native implementation makes use of a stable comparison-free linear time sort and thus takes $O(m)$ time and space. Descriptor construction is depicted in Figure 1.

### 2.2 Permutation distances

A more detailed discussion of the following is given in [16]. Recall the definition of a permutation: a bijective mapping of a finite set onto itself. This mapping $\pi$ is a member of the symmetric group $S_n$ formed by function composition on the set of all permutations of $n$ labelled objects. We write $\pi(i) = j$ to denote the action of $\pi$ with $i, j \in \{1, 2, ..., n\}$. The permutation product for $\pi_1, \pi_2 \in S_n$ is defined as function composition $\pi_1 \pi_2 = \pi_1 \circ \pi_2$, the permutation that results from first applying $\pi_2$ then $\pi_1$. Every permutation $\pi \in S_n$ can be written as a prod-

```matlab
[~, desc1] = sort(p1(:));
[~, desc2] = sort(p2(:));
distance = sum(desc1 ~= desc2);
```

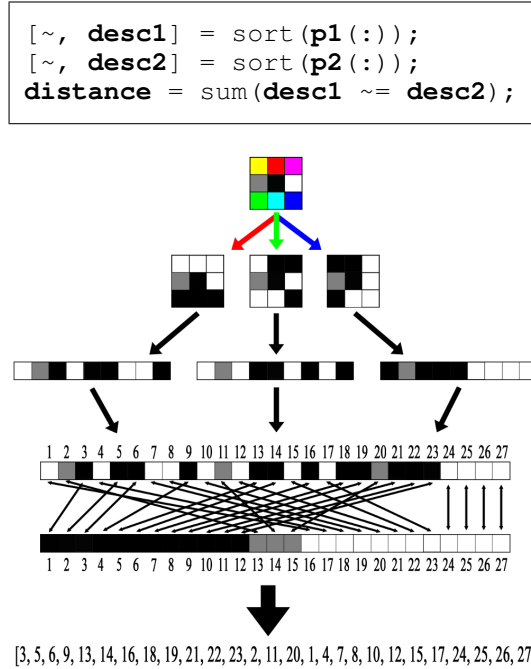

[3, 5, 6, 9, 13, 14, 16, 18, 19, 21, 22, 23, 2, 11, 20, 1, 4, 7, 8, 10, 12, 15, 17, 24, 25, 26, 27]

Figure 1: Top: LUCID feature construction and matching method in 3 lines of Matlab. *Note: ~ is used to ignore the first return value of sort; and the second value is the order permutation.* Bottom: An illustration of an image patch split into its RGB color channels, vectorized and then sorted; inducing a permutation on the indices.

uct of disjoint cycles $\sigma_1, \sigma_2, ..., \sigma_\ell$. Cycles are permutations such that $\sigma^k(i) = i$ for some $k \leq n$ where $\sigma^k = \prod_{j=1}^{k} \sigma$. We will use the notation #cycles$(\pi) = \ell$ to denote the number of cycles in $\pi$.

A convenient representation for a permutation $\pi \in S_n$ is the $n$ dimensional vector with the $i$[th] coordinate equal to $\pi(i)$; this is the *permutation vector*. The convex hull of the permutation vectors $S_n \subset \mathbb{R}^n$ is the *permutation polytope* of $S_n$. This is an $n-1$ dimensional polytope with $|S_n| = n!$ vertices. The vertices are equidistant from the centroid and lie on the surface of a circumscribed $n-1$ dimensional sphere. The vertices corresponding to two permutations $\pi_1, \pi_2 \in S_n$ are connected by an edge if they are related by a pairwise adjacent transposition. This is analogous to Kendall's tau, defined to be the minimum number of pairwise adjacent transpositions between two vectors, more precisely $K_d(\pi_1, \pi_2) = |\{(i, j) | \pi_1(i) < \pi_1(j), \pi_2(i) > \pi_2(j), 1 \leq i, j \leq n\}|$.

There are at least two classes of distances that can be defined between permutations [16]. *Spatial distances* can be viewed as measuring the distance travelled along some path between two vertices of the permutation polytope. Examples of spatial distances are Kendall's tau which steps along the edges of the polytope, the Euclidean distance which takes the straight line path, and Spearman's footrule which takes unit steps on the circumscribed sphere of the polytope. A *disorder distance* measures the disorder between two permutations and ignores the spatial structure of the polytope. Examples of disorder distances are the generalized Hamming distance $H_d(\pi_1, \pi_2) = |\{i | \pi_1(i) \neq \pi_2(i)\}|$ which is the number of elements that differ between two permutation vectors and the Cayley

distance $C_d(\pi_1, \pi_2) = n - \#\text{cycles}(\pi_2\pi_1^{-1})$ which is the minimum number of unrestricted transpositions between $\pi_1$ and $\pi_2$. We choose the generalized Hamming distance to relate our descriptors because it is much simpler than the Cayley distance to compute. Hamming also lends itself to SIMD parallel processing unlike Cayley which is inherently serial. However, if time is not a constraint experimental results show that the Cayley distance should be preferred for accuracy.

Disorder distances are not sensitive to Gaussian noise, but are highly sensitive to impulse noise. In contrast, Kendall's tau is confused by Gaussian noise, but is more resilient to impulse noise [2, 20, 18]. Impulse noise can severely corrupt these permutations since it can cause pixels in a patch to become maximal or minimal elements changing each element in the permutation vector. In the presence of moderate impulse noise the Cayley and Hamming distances will likely become maximal while Kendall's tau would be at $O(1/n)$ its maximal distance. Generally, modern imaging devices do not suffer from severe impulse noise, but there are other sources of impulse noise such as occlusions and partial shadows. LUCID is used with sparse interest points and only individual image patches would be affected by impulse noise. Since impulse noise would cause the distance to become maximal these bad matches can be reliably identified via threshold.

Kendall's tau is normally used in situations where multiple independent judges are ranking sets or subsets of objects, such as top-k lists, movie preferences or surveys. In these scenarios multiple judges are asked to rank preferences and the permutation polytope can be used as a discrete analog to histograms to gain valuable insight into the distribution of the judges' preferences. In the context of sparse image patch matching, the imaging sensor ideally acts as a single consistent judge; thus a single image patch will correspond to one vertex on the permutation polytope. Ideally, for a pair of corresponding patches in different images the permutations should be identical. Thus in our scenario the image sensor can be viewed as one judge comparing nearly identical objects. The structure of the permutation polytope becomes less important in this context.

Since the Cayley and Hamming distances are computed in linear time rather than quadratic time like Kendall's tau, they may be better suited for fast image patch matching. In section 3 we present a proof demonstrating that BRIEF is a locality sensitive hashing scheme on Kendall's tau metric between vectors of pixel intensities.

## 2.3 An efficient implementation

Table 1: Time in milliseconds to construct 10,000 descriptors and to exhaustively match 5000x5000 descriptors.

| Descriptor | Dimension | Construction | Matching |
|---|---|---|---|
| LUCID-8-Gray | 64 | 20 | 240 |
| LUCID-16-Gray | 256 | 30 | 880 |
| BRIEF | 256 | 40 | 2130 |
| LUCID-24-RGB | 1728 | 50 | 4120 |
| SURF | 64 | 450 | 420 |

Our choice to use the Hamming distance is inspired by the new Streaming SIMD Extensions (SSE) instructions. SSE is a simple way to add parallelism to native programs through vector operations. In our implementation we use a 128-bit packed comparison which gives LUCID 16x matching parallelism for grayscale image patches up to 16x16, and 8x parallelism for RGB image patches up to 147x147. Many mobile processors have these types of instructions, but even when they are not available it is still possible to gain parallelism. One additional bit per descriptor element can be reserved allowing the use of binary addition and bit masks to produce a packed Hamming distance. For descriptor lengths less than $2^{15}$, 16 bits per element are needed. This strategy supports RGB image patches up to 105x105 pixels and yields 4x parallelism on 64-bit processors. It is also possible to randomly sample a small subset of pixels before sorting to achieve greater speed. This operation can be interpreted as randomly projecting the descriptors into a lower dimension.

Order permutations are fast to construct and access memory in sequential order. Since pixel intensities are represented with small positive integers they are ideal candidates for stable linear time sorting methods like counting and radix sort. These sorting algorithms access memory in linear order and thus with the fewest number of possible cache misses. BRIEF accesses larger portions of memory than LUCID in a non-linear fashion and should incur more time consuming cache misses. Therefore LUCID offers a modest improvement in terms of descriptor construction time as shown in Table 1.

We investigate three versions of LUCID since they are the first three multiples of eight: LUCID-24-RGB, LUCID-16-Gray, and LUCID-8-Gray which respectively are LUCID on image patches that are 24x24 in RGB color, 16x16 grayscale and 8x8 grayscale. Before construction a 5x5 averaging blur is applied to the entire image to remove noise that may perturb the order permutation. BRIEF also performs pre-smoothing; Calonder et al. reported that they found a 9x9 blurring kernel to be "necessary and sufficient" [4].

We compare the running time of LUCID to the OpenCV implementations of SURF and BRIEF with default parameters on a 2.66GHz Intel® Core® i7.[1] In Table 1 timing results for SURF, BRIEF and the variants of LUCID are shown. BRIEF uses 48x48 image patches and produces a descriptor with 256 dimensions which is equal to the dimension of LUCID-16-Gray. Surprisingly, LUCID-16-Gray is faster to match than BRIEF; this was not expected since BRIEF has the same complexity as LUCID to match. This might indicate that there are further optimizations that can be made for OpenCV's implementation.

## 3   Understanding BRIEF and related methods

In [4] Calonder et al. propose BRIEF, an efficient binary descriptor. BRIEF is intended to be simple to compute and match based solely on sparse intensity comparisons. These comparisons provide for the efficient construction of a compact descriptor. Here we discuss their method as presented in [4]. Define a test $\tau$

$$\tau(\mathbf{p}; \mathbf{x}, \mathbf{y}) := \begin{cases} 1, & \text{if } \mathbf{p}(\mathbf{x}) < \mathbf{p}(\mathbf{y}) \\ 0, & \text{otherwise} \end{cases} \tag{1}$$

where $\mathbf{p}$ is a square image patch and $\mathbf{p}(\mathbf{x})$ is the smoothed value of the pixel with the local coordinates $\mathbf{x} = (u, v)^{\top}$. This test will represent one bit in the final descriptor. To construct a BRIEF descriptor a set of pre-defined pixel comparisons are performed. This pattern is a set of $n_d$ pixel coordinate pairs $(\mathbf{x}, \mathbf{y})$ that should be compared in each image patch. A descriptor is then defined to be the $n_d$ dimensional bitstring $f_{n_d}(\mathbf{p}) := \sum_{1 \le i \le n_d} 2^{i-1} \tau(\mathbf{p}; \mathbf{x_i}, \mathbf{y_i})$. Calonder et al. suggest that intuitively these pairwise intensity comparisons capture the signs of intensity gradients. However, this is not precise and in the next section we prove that the reason BRIEF works is that it inadvertently approximates Kendall's tau.

### 3.1   BRIEF is LSH on Kendall's Tau

Consider a version of BRIEF where the pixel sampling pattern consists of all $\binom{m}{2}$ pairs of pixels. Then the Hamming distance between two of these BRIEF descriptors is equivalent to the Kendall's tau distance between the pixel intensities of the vectorized image patches. The original formulation of BRIEF is LSH on the normalized Kendall's tau correlation metric.

*Proof.* Let $\mathbf{p_1}, \mathbf{p_2}$ be $m$ dimensional vectorized image patches. Define $B_k(i, j) := I(\mathbf{p_k}(i) < \mathbf{p_k}(j))$ where $I$ is the indicator function. For image patches containing $m$ pixels, BRIEF chooses a pattern of pairs $P \subseteq \{(i, j) | 1 \le i < j \le m\}$, and for two vectorized image patches $\mathbf{p_1}, \mathbf{p_2}$, it returns the score $\sum_{(i,j) \in P} I(B_1(i, j) \ne B_2(i, j))$. When $P = \{(i, j) | 1 \le i < j \le m\}$, this is precisely $K_d(\mathbf{p_1}, \mathbf{p_2})$. It can be shown that BRIEF satisfies the definition of LSH as defined in [6], consider a random pair $(i, j)$ with $i < j$. Then

$$P[B_1(i, j) \ne B_2(i, j)] = \sum_{i' < j'} \frac{1}{\binom{m}{2}} I(B_1(i', j') \ne B_2(i', j')) = K_{d_N}(\mathbf{p_1}, \mathbf{p_2}).$$

$\square$

### 3.2   The DAG of Possible Comparisons

The motivation behind BRIEF was to create a compact descriptor that could take advantage of SSE. This was in part inspired by hashing schemes that produced binary descriptors related by the Hamming distance [4]. However, these schemes require first constructing a large descriptor and then

sampling from it. BRIEF is more efficient than these methods because it skips the step of constructing the large descriptor. BRIEF is essentially a short cut and instead it immediately performs LSH. To our knowledge, the fact that BRIEF itself is an LSH scheme has not been previously discussed in the literature.

In this instance the large descriptor would be the set of all possible pairwise pixel intensity comparisons in a patch, which has an impractical $\binom{m}{2} = O(m^2)$ dimension. This set of comparisons can be modelled as a directed acyclic graph (DAG) with $m$ nodes, one for each pixel in the vectorized image patch, and $\binom{m}{2}$ edges. In this model, there exists a directed edge $(i, j)$ connecting the node that correspond to the pixel with index $i$ to the one with index $j$ in the vectorized image patch if $\mathbf{p}(i) < \mathbf{p}(j)$ where $\mathbf{p}$ is the $m$ dimensional vectorized image patch and $i \neq j$.

The topological sort of this DAG produces a unique Hamiltonian path from the sole source node to the sole sink node. The order in which the nodes are visited on this path is equivalent to the order permutation produced by a stable sort of the pixel intensities. Since this path is unique the order permutation implicitly captures all $O(m^2)$ possible comparisons in $O(m)$ space. This is possible because of the transitive property of the binary comparison and the stable order in which pixels are sorted. This is how LUCID captures all the comparative information in a patch.

In [4] Calonder et al. explored several different types of pixel sampling patterns and concluded that random sampling works the best in practice. This makes sense since BRIEF can be interpreted as randomly sampling edges from the DAG. Random sampling will eventually converge to a complete representation of the DAG through the transitive property. BRIEF can alternatively be viewed as a random projection of the adjacency matrix of the DAG.

Several variants and extensions of BRIEF have been proposed where different patterns as well as rotation and scale normalization are considered [12, 14, 19]. It follows from the proof in section 3.1 and the DAG model that these methods are dimensionality reduction schemes on Kendall's tau.

## 4 Experiments

Table 2: *Recognition Rates.* The FAST (FKD) and SURF (SKD) keypoint detectors are used to find the top 500 of 1500 keypoints in the first image for each pair. Ground truth homographies are used to warp the keypoints into the other images. The ratio of correct matches for each descriptor to the total number of ground truth matches is defined to be the recognition rate. For each image pair and keypoint detector the highest and second highest recognition rates are **bolded** with the second highest rate prefixed by an asterisk.

| Image Pair | LUCID-24-RGB | | LUCID-16-Gray | | LUCID-8-Gray | | BRIEF | | SURF | |
|---|---|---|---|---|---|---|---|---|---|---|
| — | FKD | SKD | FKD | SKD | FKD | SKD | FKD | SKD | FKD | SKD |
| Bikes 1\|2 | **0.94** | **0.92** | *0.90 | 0.83 | 0.79 | 0.54 | *0.90 | *0.90 | 0.23 | 0.75 |
| Bikes 1\|4 | *0.65 | *0.61 | 0.50 | 0.46 | 0.26 | 0.22 | **0.81** | **0.84** | 0.04 | 0.59 |
| Bikes 1\|6 | *0.19 | 0.22 | 0.13 | 0.11 | 0.07 | 0.06 | **0.73** | **0.75** | 0.01 | *0.39 |
| Wall 1\|2 | *0.54 | 0.47 | 0.38 | 0.32 | 0.21 | 0.12 | **0.87** | **0.82** | 0.17 | *0.56 |
| Wall 1\|4 | *0.16 | 0.15 | 0.12 | 0.10 | 0.08 | 0.06 | **0.64** | **0.64** | 0.11 | *0.32 |
| Wall 1\|6 | *0.04 | 0.03 | 0.03 | 0.02 | 0.02 | 0.02 | **0.17** | **0.17** | 0.03 | *0.09 |
| Light 1\|2 | **0.86** | *0.89 | *0.90 | **0.91** | 0.87 | 0.73 | 0.60 | 0.83 | 0.48 | 0.81 |
| Light 1\|4 | *0.71 | 0.75 | **0.82** | *0.76 | 0.62 | 0.55 | 0.60 | **0.79** | 0.41 | 0.71 |
| Light 1\|6 | 0.56 | 0.57 | **0.61** | 0.58 | 0.44 | 0.36 | *0.59 | **0.78** | 0.32 | *0.65 |
| Trees 1\|2 | *0.44 | *0.37 | 0.34 | 0.25 | 0.17 | 0.14 | **0.79** | **0.69** | 0.10 | 0.36 |
| Trees 1\|4 | *0.20 | 0.10 | 0.11 | 0.03 | 0.05 | 0.02 | **0.67** | **0.42** | 0.00 | *0.16 |
| Trees 1\|6 | *0.09 | 0.06 | 0.05 | 0.03 | 0.03 | 0.02 | **0.63** | **0.42** | 0.00 | *0.07 |
| Jpeg 1\|2 | 0.95 | **0.99** | *0.97 | 0.99 | 0.99 | 0.94 | 0.80 | *0.92 | 0.77 | *0.95 |
| Jpeg 1\|4 | *0.88 | 0.89 | **0.94** | **0.93** | 0.86 | 0.71 | 0.80 | *0.92 | 0.48 | 0.89 |
| Jpeg 1\|6 | *0.37 | 0.39 | 0.37 | 0.35 | 0.24 | 0.14 | **0.79** | **0.90** | 0.10 | *0.61 |

We use a subset of the commonly used benchmarking dataset used in [17].[2] Our subset consists of the image pairs that do not undergo extreme affine warping since neither BRIEF nor LUCID

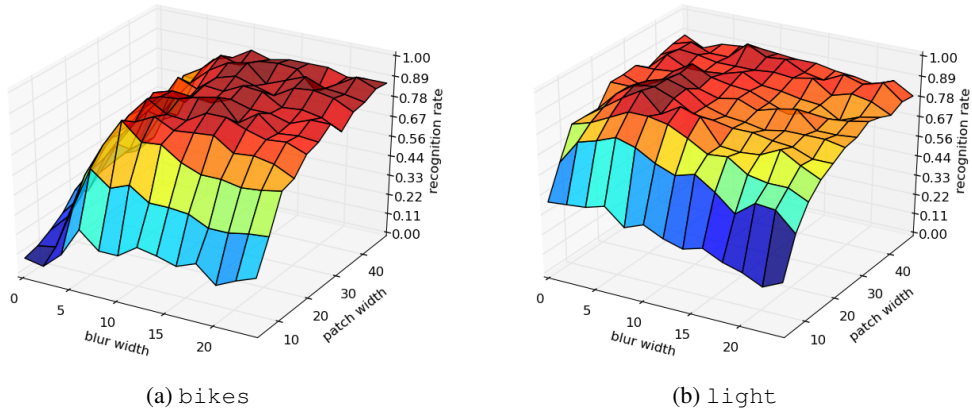

(a) `bikes`  (b) `light`

Figure 3: Recognition rates for LUCID-*-RGB on the *bikes* 1|4 and *light* 1|4 image pairs. The rates are plotted as a function of the width of the descriptor patch and of the blur kernel applied. The best 100 of 300 FAST keypoints were detected in the first image of each pair. We found a blur width of 5 rarely hurts performance and often helps. Performance increases monotonically with patch size with diminishing returns after 30x30.

account for these transformations. These image pairs are denoted by *name* 1|$k$, where $k$ represents the second image used in the pair, e.g. bikes 1|$k$ indicates the pair consisting of the first image of the bikes set to the fifth. In each experiment we detect a large number of keypoints in the first image of a set and select the top $N$ keypoints sorted by response. For each pair of images the keypoints are warped into the second image using a ground truth homography. Points that are warped out of bounds are culled before describing the points with each descriptor. Exhaustive nearest neighbor search is used to bring the points into correspondence. The *recognition rate* is then recorded as the ratio of correct matches to the number of ground truth matches.

In Table 2 we summarize the result of our comparison to BRIEF and SURF. BRIEF and LUCID perform well in most instances, though BRIEF degrades more slowly with respect to image transformations. This robustness can be attributed to the fact that BRIEF sparsely samples pixels. Most of the images are taken parallel to the horizon so orientation estimation does not help and in fact degrades SURF's performance relative to BRIEF and LUCID. LUCID performs the best on the light set which undergo exposure changes. This makes sense since the order permutation is invariant to monotonic intensity transformations and unlike BRIEF captures all the comparative information.

## 4.1 Parameter selection

LUCID has three parameters, blur kernel width, image patch size, and the option to use color or grayscale images. Figure 3 gives plots of recognition rate as a function of blur kernel width and patch size for the medium difficulty warps of two different image sets. These plots indicate that LUCID performs well with a 5x5 averaging blur kernel, and that larger patches help with diminishing returns. Though not shown here, we find that using color improves recognition performance with an expected slow down.

## 4.2 Distance distributions

Here we examine the discriminative capability of three distances, the Cayley distance, the generalized Hamming distance and Kendall's tau on pixel intensities. The Hamming distance represents LUCID which approximates the

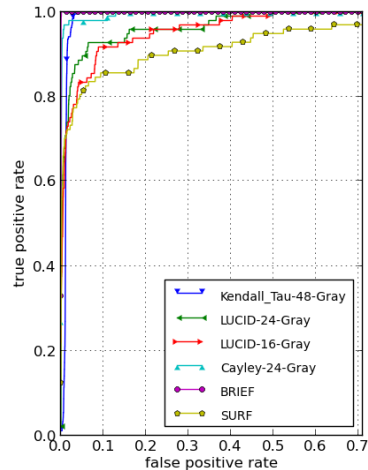

Figure 2: ROC curves for descriptors on image pair *bikes* 1|4 for 200 keypoints.

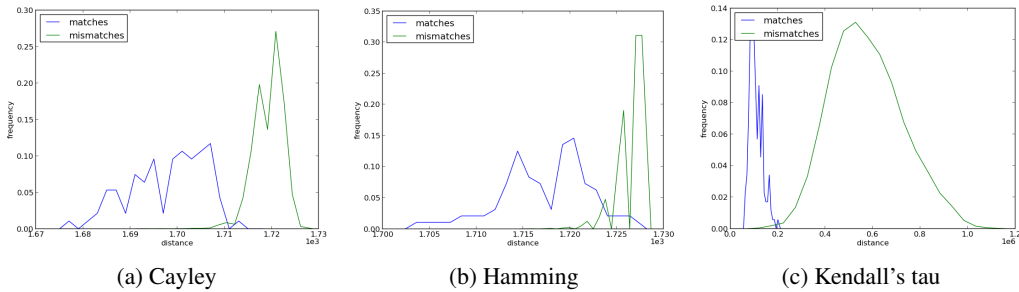

| (a) Cayley | (b) Hamming | (c) Kendall's tau |

Figure 4: Histograms of distances for correct matches and impostors for the *bikes* 1|4 image pair. The plots show the Cayley, the Hamming distance on the order permutations, and Kendall's tau on pixel intensities. These plots present a good separation of correct matches and impostors. Kendall's tau requires $O(m^2)$ time to compute while the Cayley and Hamming distances run in $O(m)$ time making them efficient alternatives. The Hamming distance is embarrassingly parallel and lends itself well to existing SSE instructions making it the most efficient distance.

Cayley distance. In Figure 4 we plot the distance distributions for correct matches and impostors, focusing on the medium difficulty warp of the *bikes* images. We chose this image set because *bikes* is a natural man-made scene and its distributions are representative for the other image sets. An ROC curve is shown in Figure 2 to visualize these results in a different way as well as for SURF and BRIEF. BRIEF does particularly well on this image pair because the only transformation that occurs is blur. Interestingly, BRIEF outperforms Kendall's Tau and the other methods that use all the pixels. BRIEF is in essence random projection dimensionality reduction for Kendall's tau. This indicates that random projections may improve the performance of the Cayley and Hamming distances as well. It is important to note that Kendall's tau is inefficient to compute with quadratic running time contrasted with the linear running time of the Cayley and generalized Hamming distances.

## 5 Conclusions and future work

In this work we have presented an analysis of BRIEF and related methods providing a theoretical basis as to how and why they work. We introduced a new simple and effective image descriptor that performs comparably to SURF and BRIEF. For our comparison and simplicity we made use of every pixel in an image patch. However, given BRIEF's superior performance to Kendall's tau we plan to explore sampling patterns of pixels and other dimensionality reduction techniques. In addition, we plan to incorporate scale and rotation normalization as in [12] and [19]. This will allow an in depth comparison of our method to descriptors like SIFT and SURF.

LUCID offers a new simplified approach for efficient feature construction and matching. We plan to investigate approximate nearest neighbor approaches like LSH and metric trees to improve the speed of matching. It would also be useful to find a binary representation of LUCID to allow for a more compact descriptor and use of existing LSH schemes. It is already possible to obtain such a representation for LUCID through a method like WTAHash [23]. WTAHash produces an embedding for ordinal feature spaces such that transformed feature vectors are in binary form and the Hamming distance between them closely approximates the original metric.

Finally, we hope that this new understanding of BRIEF and other binary descriptors will allow for the creation of new efficient visual feature descriptors. Spending less time processing visual features provides more CPU time for core functionality and application complexity enabling new real-time applications.

## 6 Acknowledgements

We would like to acknowledge Brian McFee for his helpful conversations. This work was supported by ONR MURI Grant #N00014-08-1-0638.

## Footnotes

[1]We used a stable release of OpenCV, version 2.4.3. OpenCV is open source and all versions are publicly available at `http://opencv.willowgarage.com`.

[2]The dataset is available for download at `http://www.robots.ox.ac.uk/~vgg/research/affine/`

# References

[1] Herbert Bay, Andreas Ess, Tinne Tuytelaars, and Luc Van Gool. Speeded-up robust features (SURF). *Comput. Vis. Image Underst.*, 110(3):346–359, June 2008.

[2] D.N. Bhat and S.K. Nayar. Ordinal measures for image correspondence. *Pattern Analysis and Machine Intelligence*, 20(4):415–423, Apr 1998.

[3] A. Bosch, A. Zisserman, and X. Muoz. Image classification using random forests and ferns. In *Computer Vision, 2007. ICCV 2007*, pages 1–8, Oct. 2007.

[4] Michael Calonder, Vincent Lepetit, Christoph Strecha, and Pascal Fua. Brief: binary robust independent elementary features. In *Proceedings of the 11th European conference on Computer vision: Part IV*, ECCV'10, pages 778–792, Berlin, Heidelberg, 2010. Springer-Verlag.

[5] A. Cayley. Lxxvii. note on the theory of permutations. *Philosophical Magazine Series 3*, 34(232):527–529, 1849.

[6] Moses S. Charikar. Similarity estimation techniques from rounding algorithms. In *Proceedings of the thiry-fourth annual ACM symposium on Theory of computing*, STOC '02, pages 380–388, New York, NY, USA, 2002.

[7] Michael Deza, Liens ecole Normale Suprieure, and Tayuan Huang. Metrics on permutations, a survey. *Journal of Combinatorics, Information and System Sciences*, 1998.

[8] Persi Diaconis and R. L. Graham. Spearman's footrule as a measure of disarray. *Journal of the Royal Statistical Society. Series B (Methodological)*, 39(2):pp. 262–268, 1977.

[9] M. A. Fligner and J. S. Verducci. Distance based ranking models. *Journal of the Royal Statistical Society. Series B (Methodological)*, 48(3):pp. 359–369, 1986.

[10] R. W. Hamming. Error detecting and error correcting codes. *Bell System Technical Journal*, 29(2):147–160, 1950.

[11] M. G. Kendall. A new measure of rank correlation. *Biometrika*, 30(1/2):pp. 81–93, 1938.

[12] S Leutenegger, M Chli, and R Siegwart. BRISK: Binary robust invariant scalable keypoints. In *Proc. of the IEEE International Conference on Computer Vision (ICCV)*, 2011.

[13] Bing Li, Rong Xiao, Zhiwei Li, Rui Cai, Bao-Liang Lu, and Lei Zhang. Rank-SIFT: Learning to rank repeatable local interest points. In *Computer Vision and Pattern Recognition (CVPR)*, pages 1737–1744, June 2011.

[14] Jie Liu and Xiaohui Liang. I-BRIEF: A fast feature point descriptor with more robust features. In *Signal-Image Technology and Internet-Based Systems (SITIS)*, pages 322–328, Dec. 2011.

[15] D.G. Lowe. Object recognition from local scale-invariant features. In *Computer Vision*, volume 2, pages 1150–1157, 1999.

[16] John I. Marden. *Analyzing and Modeling Rank Data*. Chapman & Hall, 1995.

[17] Krystian Mikolajczyk and Cordelia Schmid. A performance evaluation of local descriptors. *IEEE Trans. Pattern Anal. Mach. Intell.*, 27(10):1615–1630, October 2005.

[18] A. Mittal and V. Ramesh. An intensity-augmented ordinal measure for visual correspondence. In *Computer Vision and Pattern Recognition*, volume 1, pages 849–856, June 2006.

[19] Ethan Rublee, Vincent Rabaud, Kurt Konolige, and Gary Bradski. ORB: An efficient alternative to SIFT or SURF. *International Conference on Computer Vision*, 95(1):2564–2571, 2011.

[20] S Scherer, P Werth, and A Pinz. The discriminatory power of ordinal measures – towards a new coefficient. 1, 1999.

[21] Timothy B. Terriberry, Lindley M. French, and John Helmsen. GPU accelerating speeded-up robust features. In *Proceedings of the 4th International Symposium on 3D Data Processing, Visualization and Transmission*, 3DPVT '08, pages 355–362, Atlanta, GA, USA, 2008.

[22] M. Toews and W. Wells. Sift-rank: Ordinal description for invariant feature correspondence. In *Computer Vision and Pattern Recognition*, pages 172–177, June 2009.

[23] Jay Yagnik, Dennis Strelow, David A. Ross, and Ruei-Sung Lin. The power of comparative reasoning. In *ICCV*, pages 2431–2438, 2011.

